# On a Connection between Importance Sampling and the Likelihood Ratio Policy Gradient

**Jie Tang and Pieter Abbeel**
Department of Electrical Engineering and Computer Science
University of California, Berkeley
Berkeley, CA 94709
{jietang, pabbeel}@eecs.berkeley.edu

## Abstract

Likelihood ratio policy gradient methods have been some of the most successful reinforcement learning algorithms, especially for learning on physical systems. We describe how the likelihood ratio policy gradient can be derived from an importance sampling perspective. This derivation highlights how likelihood ratio methods under-use past experience by (i) using the past experience to estimate *only* the gradient of the expected return $U(\theta)$ at the current policy parameterization $\theta$, rather than to obtain a more complete estimate of $U(\theta)$, and (ii) using past experience under the current policy *only* rather than using all past experience to improve the estimates. We present a new policy search method, which leverages both of these observations as well as generalized baselines—a new technique which generalizes commonly used baseline techniques for policy gradient methods. Our algorithm outperforms standard likelihood ratio policy gradient algorithms on several testbeds.

## 1  Introduction

Policy gradient methods have been some of the most effective learning algorithms for dynamic control tasks in robotics. They have been applied to a variety of complex real-world reinforcement learning problems, such as hitting a baseball with an articulated arm robot [1], constrained humanoid robotic motion planning [2], and learning gaits for legged robots [3, 4, 5]. For such robotics tasks real-world trials are typically the most time consuming factor in the learning process. Making efficient use of limited experience is crucial for good performance.

In this paper we describe a novel connection between likelihood ratio based policy gradient methods and importance sampling. Specifically, we show that the likelihood ratio policy gradient estimate is equivalent to the gradient of an importance sampled estimate of the expected return function estimated using only data from the current policy. This insight indicates that likelihood ratio policy gradients are quite naive in terms of data use, and suggests an opportunity for novel algorithms which use all past data more efficiently by working with the importance sampled expected return function directly.

Our main contributions are as follows. First, we develop algorithms for global search over the importance sampled expected return function, allowing us to make more progress for a given amount of experience. Our approach uses estimates of the importance sampling variance to constrain the search in a principled way. Second, we derive generalizations of optimal policy gradient baselines which are applicable to the importance sampled expected return function.

Section 2 describes preliminaries on Markov decision processes (MDPs), policy gradient methods and importance sampling. Section 3 describes the novel connection between importance sampling and likelihood ratio policy gradients, and Section 4 examines our novel minimum variance baselines. Section 5 outlines our proposed method. Section 6 relates our method to prior work. Section 7 demonstrates the effectiveness of the proposed methods on standard reinforcement learning testbeds.

## 2 Preliminaries

**Markov Decision Processes.** A Markov decision process (MDP) is a tuple $(S, A, T, R, D, \gamma, H)$, where S is a set of states; $A$ is a set of actions/inputs; $T = \{P(\cdot|s, u)\}_{s,u}$ is a set of state transition probabilities ($P(\cdot|s, u)$ is the state transition distribution upon taking action $u$ in state $s$); $R : S \times A \mapsto \mathbb{R}$ is the reward function; $D$ is a distribution over states from which the initial state $s_0$ is drawn; $0 < \gamma < 1$ is the discount factor; and $H$ is the horizon time of the MDP, so that the MDP terminates after $H$ steps[1]. A policy $\pi$ is a mapping from states $S$ to a probability distribution over the set of actions $A$. We will consider policies parameterized by a vector $\theta \in \mathbb{R}^n$. We denote the expected return of a policy $\pi_\theta$ by

$$U(\theta) = \mathrm{E}_{P(\tau;\theta)}\left[\sum_{t=0}^{H} \gamma^t R(s_t, u_t)|\pi_\theta\right] = \sum_\tau P(\tau; \theta)R(\tau). \tag{2.1}$$

Here $P(\tau; \theta)$ is the probability distribution induced by the policy $\pi_\theta$ over all possible state-action trajectories $\tau = (s_0, u_0, s_1, u_1, \ldots, s_H, u_H)$. We overload notation and let $R(\tau) = \sum_{t=0}^{H} \gamma^t R(s_t, u_t)$ be the (discounted) sum of rewards accumulated along the state-action trajectory $\tau$.

**Likelihood Ratio Policy Gradient.** Likelihood ratio policy gradient methods perform a (stochastic) gradient ascent over the policy parameter space $\Theta$ to find a local optimum of $U(\theta)$. One well-known technique called REINFORCE [6, 7] expresses the gradient $\nabla_\theta U(\theta)$ as follows:

$$g = \nabla_\theta U(\theta) = \mathrm{E}_{P(\tau;\theta)}[\nabla_\theta \log P(\tau; \theta)R(\tau)] \approx \hat{g} = \frac{1}{m}\sum_{i=1}^{m} \nabla_\theta \log P(\tau^{(i)}; \theta)R(\tau^{(i)}),$$

where the rightmost expression provides us an unbiased estimate of the policy gradient from $m$ sample paths $\{\tau^{(1)}, \ldots, \tau^{(m)}\}$ obtained from acting under policy $\pi_\theta$. Using the Markov assumption, we can decompose $P(\tau; \theta)$ into a product of conditional probabilities and we obtain $\nabla_\theta \log P(\tau^{(i)}; \theta) = \sum_{t=0}^{H} \nabla_\theta \log \pi_\theta(u_t^{(i)}|s_t^{(i)})$. Hence no access to a dynamics model is required to compute an unbiased estimate of the policy gradient. REINFORCE has been shown to be moderately efficient in terms of number of samples used [6, 7].

To reduce the variance it is common to use baselines. Since $\mathrm{E}_P[\nabla_\theta \log P(\tau; \theta)] = \nabla_\theta \sum_\tau P(\tau; \theta) = \nabla_\theta 1 = \mathbf{0}$ we can add $b^\top \nabla_\theta \log P(\tau; \theta)$ (where $b$ is a vector which can be optimized to minimize variance) to the REINFORCE gradient estimate without biasing it [8, 9]. Past work often used a scalar $b$, resulting in:

$$\nabla_\theta U(\theta) = \mathrm{E}_{P(\tau;\theta)}[\nabla_\theta \log P(\tau; \theta)(R(\tau) - b)] \approx \hat{g} = \frac{1}{m}\sum_{i=1}^{m} \nabla_\theta \log P(\tau^{(i)}; \theta)(R(\tau^{(i)}) - b).$$

**Importance Sampling.** For a general function $f$ and a probability measure $P$, computing a quantity of interest of the form

$$\mathrm{E}_{P(X)}[f(X)] = \int_x P(x)f(x)dx.$$

can be computationally challenging. The expectation is often approximated with a sample-based estimate. However, samples from $P$ could be difficult to obtain, or $P$ might have very low probability where $f$ takes its largest values. Importance sampling provides an alternative solution which uses samples from a different distribution $Q$. Given samples from $Q$, we can estimate the expectation w.r.t. $P$ as:

$$\begin{aligned}\mathrm{E}_{P(X)}[f(X)] &= \mathrm{E}_{Q(X)}\left[\frac{P(X)}{Q(X)}f(X)\right]\\ &\approx \frac{1}{m}\sum_{i=1}^{m} \frac{P(x^{(i)})}{Q(x^{(i)})}f(x^{(i)}) \text{ with } x^{(i)} \sim Q\end{aligned}$$

In the above, we assume $Q(x) = 0 \Rightarrow P(x) = 0$. Hence, one can sample from a different distribution $Q$ and then simply re-weight the samples to obtain an unbiased estimate. This can be readily leveraged to estimate the expected return of a stochastic policy [10] as follows:

$$\widehat{U}(\theta) = \frac{1}{m}\sum_{i=1}^{m} \frac{P(\tau^{(i)};\theta)}{Q(\tau^{(i)})}R(\tau^{(i)}), \quad \tau^{(i)} \sim Q \tag{2.2}$$

where we assume $Q(\tau) = 0 \Rightarrow P(\tau; \theta) = 0$. If we choose $Q(\tau) = P(\tau; \theta')$, then we are estimating the return of a policy $\pi_\theta$ from sample paths obtained from acting according to a policy $\pi_{\theta'}$. Evaluating the importance weights does not require a dynamics model: $\frac{P(\tau^{(i)};\theta)}{P(\tau^{(i)};\theta')} = \frac{\prod_{t=0}^{H} \pi_\theta(u_t|s_t)}{\prod_{t=0}^{H} \pi_{\theta'}(u_t|s_t)}$. If we have samples from many different distributions $P(\tau; \theta^{(j)})$, a standard technique is to create a fused empirical distribution $Q(\tau) = \frac{1}{m}\sum_{j=1}^{m} P(\tau; \theta^{(j)})$ to enable use of all past data [10].

# 3 Likelihood Ratio Policy Gradient via Importance Sampling

We now outline a novel connection between policy gradients and importance sampling. A set of trajectories $\{\tau^{(1)}, \ldots, \tau^{(m)}\}$ sampled from policy $\pi_{\theta^*}$ induces a distribution over paths $Q(\tau) = P(\tau; \theta^*)$. Let $\widehat{U}(\theta^*)$ denote the importance sampled estimate of $U(\theta)$ at $\theta^*$. Using Equation (2.2), we have:

$$
\begin{aligned}
\frac{\partial \widehat{U}}{\partial \theta_j}(\theta^*) &= \frac{1}{m} \sum_{i=1}^{m} \frac{1}{Q(\tau^{(i)})} \frac{\partial P(\tau^{(i)}; \theta^*)}{\partial \theta_j} R(\tau^{(i)}) \\
&= \frac{1}{m} \sum_{i=1}^{m} \frac{P(\tau^{(i)}; \theta^*)}{Q(\tau^{(i)})} \frac{\partial \log P(\tau^{(i)}; \theta^*)}{\partial \theta_j} R(\tau^{(i)}) \\
&= \frac{1}{m} \sum_{i=1}^{m} \frac{\partial \log P(\tau^{(i)}; \theta^*)}{\partial \theta_j} R(\tau^{(i)}) \quad \text{(using } Q(\tau) = P(\tau; \theta^*)).
\end{aligned}
\tag{3.1}
$$

Equation 3.1 is the $j$'th entry of the likelihood ratio based estimate of the gradient of $U(\theta)$ at $\theta^*$. This analysis shows that the standard likelihood ratio policy gradient can be interpreted as forming an importance sampling based estimate of the expected return based on the runs under the current policy $\pi_{\theta^*}$ and then using this estimate of the expected return function only to estimate a gradient at $\theta^*$. In doing so, it fails to make efficient use of the trials from past policies: (i) It only uses the gradient of the function $\widehat{U}(\theta)$ at the point $\theta^*$, rather than all information provided by the function $\widehat{U}(\theta)$, and (ii) It only uses the runs under the most recent policy $\pi_{\theta^*}$, rather than using a more informed importance sampling based estimate that uses all past data.

Instead of only using local information from a single policy to drive our learning, we can use *global* information provided by $\widehat{U}(\theta)$ using trials run under *all past* policies. Such importance sampling based methods (as have been proposed in [10]) should be able to learn from fewer trial runs than the currently widely popular likelihood ratio based methods.

**Generalization to G(PO)MDP / Policy Gradient Theorem formulation.** The observation that past rewards do not depend on future states or actions is leveraged by the G(PO)MDP [8] and the Policy Gradient Theorem [11] variations on REINFORCE to reduce the variance on their gradient estimates. This same observation can also be leveraged when estimating the expected return function itself. Let $\tau_{1:t}$ denote the state action sequence experienced from time $1$ through time $t$, then we have

$$
U(\theta) = \sum_{\tau} P(\tau; \theta) R(\tau) = \sum_{\tau} \sum_{t=0}^{H} P(\tau_{1:t}; \theta) R(s_t, u_t).
\tag{3.2}
$$

For simplicity of notation we will continue to describe our approach in terms of the expression for $U(\theta)$ given in Equation (2.1), but our generalization of baselines, and our policy search algorithm are equally applicable when using the expression for $U(\theta)$ we present in Equation (3.2).

# 4 Generalized Unbiased Baselines

Previous work has shown that the REINFORCE gradient estimate benefits greatly from the addition of an optimal baseline term [12, 9, 8]. In this section, we show that policy gradient baselines are special cases of a more general variance reduction technique. Our result generalizes policy gradient baselines in three ways: (i) It applies to estimating expectations of any random quantity, not just policy gradients; (ii) It allows for baseline matrices and higher-dimensional tensors, not just vectors; and (iii) It can be applied recursively to yield baseline terms for baselines since baselines are themselves expectations.

**Minimum Variance Unbiased Baselines.** Given a random variable $X \sim P_\theta(X)$, where $P_\theta$ is a parametric probability distribution with parameter $\theta$, we have that $\mathrm{E}_{P_\theta}[\nabla_\theta \log P_\theta(X)] = 0$. Hence for any constant vector $b$ and any scalar function $h(X)$, we have that $\frac{1}{m} \sum_{i=1}^{m} (h(x^{(i)}) - b^\top \nabla_\theta \log P_\theta(x^{(i)}))$ with $x^{(i)}$ drawn from $P_\theta$ is an unbiased estimator of the scalar quantity $\mathrm{E}_{P_\theta}[h(X)]$. The variance of this estimator is minimized when the variance of the random variable $g(X) = h(X) - b^T \nabla_\theta log P_\theta(X)$ is minimized. This variance is given by:

$$
\mathrm{Var}_{P_\theta}[h(X) - b^\top \nabla_\theta \log P_\theta(X)] = \mathrm{E}_{P_\theta}[(h(X) - b^\top \nabla_\theta \log P_\theta(X))^2] - (\mathrm{E}_{P_\theta}[h(X) - b^\top \nabla_\theta \log P_\theta(X)])^2.
$$

As $b^\top \mathrm{E}_{P_\theta}[\nabla_\theta \log P_\theta(X)] = 0$, the second term is independent of $b$. Setting the gradient of the first term with respect to $b$ equal to zero yields the minimum variance baseline

$$
b = \mathrm{E}_{P_\theta}[\nabla_\theta \log P_\theta(X) \nabla_\theta \log P_\theta(X)^\top]^{-1} \mathrm{E}_{P_\theta}[\nabla_\theta \log P_\theta(X) h(X)].
\tag{4.1}
$$

The baselines commonly employed with REINFORCE, GPOMDP, and other likelihood ratio policy gradient methods can be derived as special cases of this generalized baseline [12].

**Minimum Variance Unbiased Baselines with Importance Sampling.** When using importance sampling with $x^{(i)}$ drawn from $Q$, we have an unbiased estimator of the form $\frac{1}{m}\sum_{i=1}^{m}\frac{P_\theta(x^{(i)})}{Q(x^{(i)})}(h(x^{(i)}) - b^\top \nabla_\theta \log P_\theta(x^{(i)}))$ with a minimum variance baseline vector

$$b = \mathrm{E}_Q\left[\frac{P_\theta(X)}{Q(X)}\nabla_\theta \log P_\theta(X)\frac{P_\theta(X)}{Q(X)}\nabla_\theta \log P_\theta(X)^\top\right]^{-1} \mathrm{E}_Q\left[\frac{P_\theta(X)}{Q(X)}\nabla_\theta \log P_\theta(X)\frac{P_\theta(X)}{Q(X)}h(X)\right]. \quad (4.2)$$

**Baselines.** The minimum variance technique is naturally extended to vector-valued or matrix-valued random variables $h(X)$. For each entry in $h(X)$ we can compute a minimum variance baseline vector $b$ using Equation (4.1) or (4.2). In general, if $h(X)$ is an $n$-dimensional tensor, we can stack these baseline vectors into a $n+1$-dimensional tensor. Indeed, in the case of REINFORCE we would obtain a baseline matrix, rather than a baseline scalar (as in the original work [7]) and rather than a vector baseline (as described in later work, such as [12]). The baselines themselves are estimated from sample data. Using standard policy gradient methods, it can be impractical to run enough trials to accurately fit such baselines. By using importance sampling to reuse data we can use richer baseline terms in our estimators.

**Recursive Baselines.** The baselines are themselves composed of expectations. It is possible to recursively insert minimum variance unbiased baseline terms into these expectations in order to reduce the variance on the baseline estimates. However, the number of baseline parameters being estimated increases rapidly in this recursive process. Moreover, if we estimate multiple expectations from the same set of samples, these estimates become correlated and the final result is no longer unbiased. In practice, these baselines can be regularized to match the amount of available data. In Section 8 we empirically investigate the performance of several different baseline schemes.

# 5 Policy Search Using $\widehat{U}$

We propose the algorithm outlined in Figure 1. It uses importance sampling with optimal generalized baselines to obtain estimates $\widehat{U}(\theta)$ of the expected return function based on the data gathered so far. This estimator allows to search for a $\theta$ which improves the expected return. It maintains a list of candidate policy parameters from which it searches for improvements. Memory-based search allows backtracking away from unpromising parts of the search space without taking additional, costly trials on the real platform.

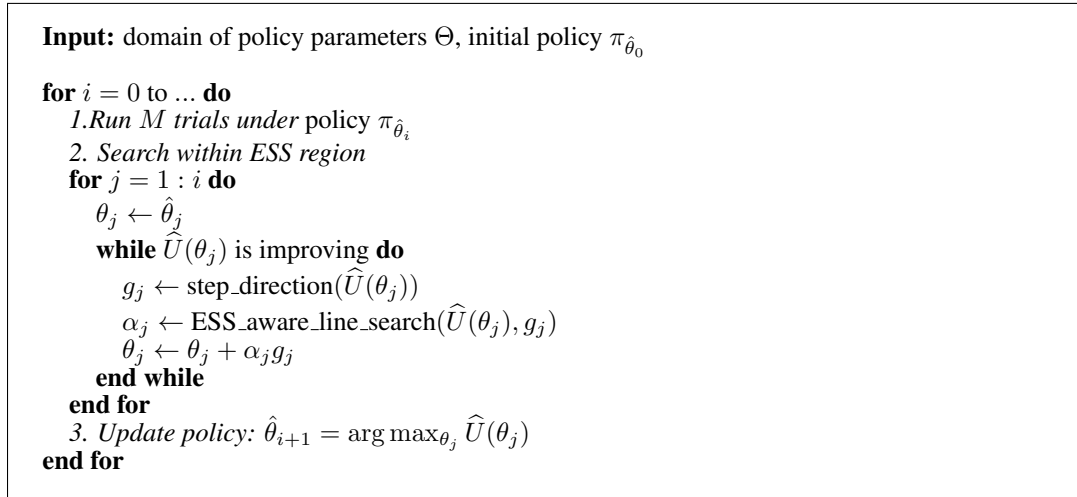

Figure 1: Our policy search algorithm.

**Estimate of Expected Returns:** We use weighted importance sampling, and add a baseline to Equation (2.2):

$$\widehat{U}(\theta) = \frac{1}{Z}\sum_{i=1}^{m}\frac{P(\tau^{(i)};\theta)}{Q(\tau^{(i)})}(R(\tau^{(i)}) - b^\top \nabla_\theta \log P(\tau^{(i)};\theta)), \ \ Z = \sum_{i=1}^{m}\frac{P(\tau^{(i)};\theta)}{Q(\tau^{(i)})}, \quad (5.1)$$

where $i$ indexes over all past trials, and $Q$ is the empirical distribution over past trials (see Section 2).

**Optimal Baseline:** Applying Equation (4.2) we get the following sample based estimate of the optimal baseline $b$ for the estimate of the expected return function:[2]

$$b = \left( \frac{1}{m} \sum_{i=1}^{m} \left( \frac{P(\tau^{(i)};\theta)}{Q(\tau^{(i)})} \right)^2 \nabla_\theta \log P_\theta(\tau^{(i)}) \nabla_\theta \log P(\tau^{(i)};\theta)^\top \right)^{-1}$$

$$\left( \frac{1}{m} \sum_{i=1}^{m} \left( \frac{P(\tau^{(i)};\theta)}{Q(\tau^{(i)})} \right)^2 \nabla_\theta \log P(\tau^{(i)};\theta) R(\tau^{(i)})) \right). \tag{5.2}$$

**ESS Search Region:** As our policy search steps away from areas of $\Theta$ where we have gathered sample data, the variance of our estimator $\widehat{U}$ increases and our function estimate becomes unreliable. The effective sample size $\text{ESS} = \frac{m}{1+\text{Var}(w_i)}$ is commonly used to measure the quality of an importance sampled estimate [13]. Here $w_i$ are the normalized importance weights and $M$ is the number of trials. Our policy search only considers parameter values $\theta$ with sufficiently high ESS.

**Step Direction:** We use the finite-difference gradient of $\widehat{U}$ as the step direction for the inner loop of the policy search. In theory, since every outer iteration searches for a local optimum within the ESS region, the choice of step direction affects only the amount of computation and not the number of trials required for convergence.[3]

**Line Search:** One issue with gradient based optimization methods is the need to choose the right step size. One solution is to use adaptive line search-based step size rules like the Armijo rule [15].[4] For traditional likelihood ratio policy search methods this would require additional trials. By contrast, no new trials are required when using importance sampling.[5]

# 6 Prior Work

Various past approaches use the idea of constructing a model of the system from sample data, which can be used to search for the optimal policy, e.g., [16], [10], [17]. In contrast to Sutton's DYNA, our method attempts to directly optimize the expected return function by varying policy parameters rather than building a model for the environment. Cao [17] also uses importance sampling to reuse past data for estimating policy gradients, but focuses on estimating local gradient information rather than global surface information. The work of Peshkin and Shelton [10] is most similar in spirit to our policy search method. They use importance sampling to construct a "proxy" environment from sampled data which can be used to evaluate the expected return at arbitrary policies. They apply a hill-climbing policy search to this "proxy" surface. This technique does not use estimates of the importance sampling variance to restrict the search, does not use generalized minimum variance baselines, and does not use memory. Our experiments show that these improvements are necessary to outperform standard policy gradient methods across our test domains.

Our general approach of estimating and optimizing the expected return function instead of the gradient of the expected return function allows for non-local policy steps. Recent EM-based policy search methods [18, 14] are able to make larger steps by optimizing a local lower bound on the expected return function. These methods can use importance sampling to make better use of data. This lower bound objective function and update step could be used in our memory based approach instead of following the finite difference gradient step.

We explained throughout the paper the relationship with earlier methods such as REINFORCE [7, 6] and GPOMDP [8, 9]. PEGASUS [19] is an efficient alternative policy search method but can only be used if a simulation model is available.

Recent work has suggested following the natural gradient direction [20, 21, 22]. The natural gradient approach is a parameterization invariant second order method which finds the direction which

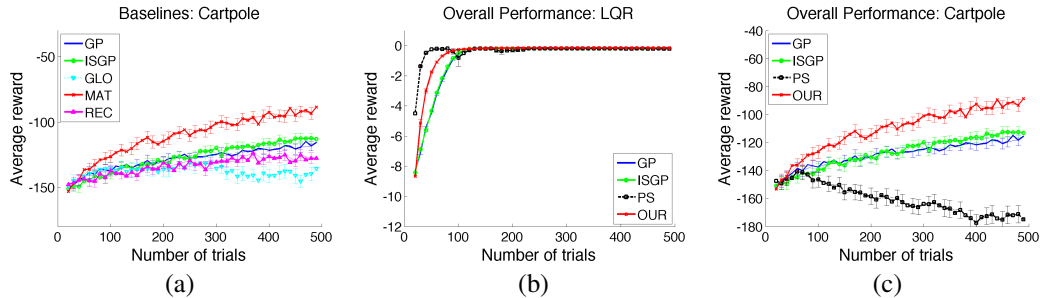

Figure 2: (a) Performance of various choices for the higher level baselines in our approach. We have a matrix baseline (MAT), and a recursive baseline (REC). For reference, we also plot our approach without an optimal baseline (GLO), GPOMDP (GP), and IS GPOMDP (ISGP). (b), (c) Performance evaluation on LQR and Cartpole. The algorithms considered are the GPOMDP likelihood ratio policy gradient method (GP), GPOMDP with importance sampling (ISGP), Peshkin and Shelton's algorithm (PS), and our approach (OUR).

maximize the ratio of the improvement of the objective function over the change in distribution over trajectories. Our approach exploits a similar intuition through consideration of variance through the effective sampling size (ESS)—preferring regions for which the past experience gives a good estimate.

Natural actor critic (NAC) approaches have enjoyed substantial success on real-life robotics tasks [1, 23]. In the episodic setting, which we consider in this paper, the only difference between episodic NAC and natural gradient is in the estimate of the baseline. Episodic NAC computes a scalar baseline by solving an LSTD-Q type regression rather than, e.g., using a minimum variance baseline criterion.[6]

## 7 Experimental Setup

We present experiments on four testbeds: LQR, cartpole, mountaincar, and acrobot. The details of each experimental testbed can be found in the appendix. Though the systems are simulated, the learning algorithms cannot make use of the simulation dynamics except by gathering trials. For each testbed we randomly generated a pool of initial policies until one is found that does not achieve the worst case return We then used our policy gradient algorithms to optimize performance. The same set of initial policies is used across learning algorithms. We focus on an analysis of performance when only allowed for a small number of trials: In each of the following experiments we run 50 iterations of policy search, running $M$ trials for each policy at each iteration.

## 8 Experimental Results

In our experimental results, we first evaluate several generalized baselines in the context of our policy search algorithm. We then break down the effectiveness of each component of our algorithm: memory based search, optimal baselines, and ESS search region. Our policy search outperform likelihood ratio methods on two of the testbeds and performs equally well on the two remaining ones. Performance is reported as the expected return versus the number of sampled trials. The expected return is plotted on the y-axis. Error bars are shown based on running each instance with 10 initial policies. The number of trials is plotted on the x-axis.

**Generalized Baseline Experiments:** There are a variety of choices in our generalized baseline technique: We can vary the dimensionality of the baseline terms to add, the depth of the recursive baseline, and what (if any) regularization to use.

We implemented our policy search using three different baseline techniques. We used a vector baseline, a matrix baseline, and a recursive tensor baseline on the matrix baseline. Figure 2 (a) shows the average reward received plotted against the number of trials run for the matrix (MAT) and recursive tensor (REC) baselines. The vector baseline was not able to improve the initial policies. The matrix baseline outperforms the other baselines and we use it going forward.

**Components of Our Approach:** Figure 3 examines each of the central contributions of our algorithm (memory based search, baselines, and ESS). We tested our approach without any of the

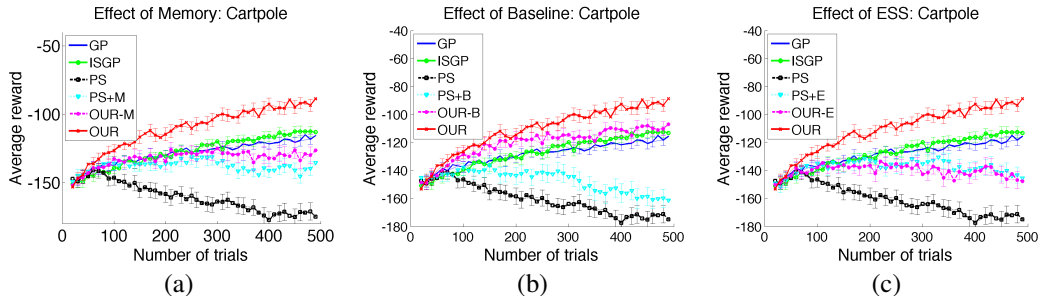

Figure 3: This figure demonstrates the effect of (a) memory based search (b) optimal baselines, and (c) ESS search region on cartpole performance. In each figure, we show the performance of Peshkin and Shelton's approach (PS) and our approach (OUR). In addition, we show the performance with memory only (PS+M), baselines only (PS+B), and ESS only (PS+E), and our approach with memory (OUR-M), baselines (OUR-B), and ESS (OUR-E) removed. GPOMDP (GP) and IS GPOMDP (ISGP) are also plotted for reference purposes.

three components, which is equivalent to Peshkin and Shelton's algorithm [10], which we label PS. We added each one of the three components individually, labeled PS+M, PS+B, PS+E. We also tested the performance with two out of three components, labeled OUR-M, OUR-B, and OUR-E respectively. Finally we tested the performance of our approach with all three components. The results indicate that each of the three components is improving performance with ESS and memory based being the most important components. Without any one of the components our approach has difficulty outperforming importance sampled GPOMDP.

**Comparison With Likelihood Ratio Policy Gradients:** We have compared several episodic likelihood ratio algorithms against our global policy search algorithm. We run $M = 10$ trials per iteration, and repeat each trial 10 times. For the likelihood ratio algorithms, we use the appropriate optimal baselines [12] and hand-tune the step size. As a comparison, we have also implemented policy gradient algorithms which use importance sampling to estimate the gradient of $\widehat{U}$. Figure 2 plots the reward received as a function of the number of real trials sampled from the system. We plot our global search approach against GPOMDP, an importance sampled GPOMDP (IS GPOMDP), and an implementation of Peshkin and Shelton's global search.[7] Our approach is consistently able to improve its initial policy, outperforming likelihood ratio policy gradient methods on both the cartpole and LQR testbeds. In general, importance sampling based methods outperform non-importance sampling based algorithms, which work poorly when given few trials. All algorithms in consideration performed poorly on the mountaincar and acrobot testbed—none of them showing significant improvement in performance through learning.

## 9   Conclusion

We have shown that policy gradient methods are a special case of gradient descent over the importance sampled expected return function $\widehat{U}$. Since our approach provides a full approximation of the expected return function, we can use global information in addition to gradient information to achieve faster learning. We have also shown that optimal baselines for standard policy gradient methods can be seen as special cases of a more general variance reduction technique. Our importance sampling approach allows us to leverage more data to fit generalized baseline terms in our estimators. Our experiments show our algorithm requires fewer trials than current policy gradient methods on several testbeds and no more trials on the remaining testbeds, making it appealing for robotic learning tasks for which trials are expensive.

**Acknowledgments**

The authors thank Jan Peters and Hamid Reza Maei for insightful discussions and the anonymous reviewers for their feedback. This work was supported in part by NSF under award IIS-0931463. Jie Tang is supported by the Department of Defense (DoD) through the National Defense Science & Engineering Graduate Fellowship (NDSEG) Program.

## Appendix

**(i) LQR:** We use the formulation given in [21]. We use a linear parameterized policy with parameters $K \in \mathbb{R}^2$, given by $u(t) \sim N(Lx(t), \sigma)$, $L = -1.999 + \frac{1.998}{1+e^{K_1}}$ and $\sigma = 0.001 + \frac{1}{1+e^{K_2}}$.[8] The initial state is drawn from $x(0) \sim N(0.3, 0.1)$, and the dynamics are given by $x(t+1) = 0.7 * x(t) + u(t) + N(0, 0.01)$. The system incurs a penalty of $-(x(t)^2 + u(t)^2)$ at each time step. Each episode was 20 time steps.

**(ii) Cartpole:** This task consists of a cart moving along a track while balancing a pole. The goal of this task is to move the cartpole back to the origin as quickly as possible while keeping the pole upright. Following the formulation given in [24], our control input is drawn from the policy $u \sim N(K^\top x, \sigma)$, with state $x = [x, \dot{x}, \theta, \dot{\theta}]$ and policy parameters $K = [K_1, K_2, K_3, K_4, \sigma]$. The dynamics are given by $\ddot{x} = \frac{F - m_p l(\ddot{\theta} \cos\theta - \dot{\theta}^2 \sin\theta)}{m_c + m_p}$, and $\ddot{\theta} = \frac{g \sin\theta(m_c + m_p) - (u_t + m_p l\dot{\theta}^2 \sin\theta)\cos\theta}{\frac{4}{3}l(m_c + m_p) - m_p l\cos^2\theta}$. Here $m_p = 0.1, m_c = 1.0, l = 0.5, g = 9.81$. The control interval was $0.02s$. We solve the dynamics using a fourth order Runge-Kutta method. We run each episode for 200 time steps, though the episode terminates once the cartpole has failed (defined as whenever $|x| > 2.4m$ or $|\theta| > 0.7rad$). The reward function is $-2$ for every time step after the failure occurs, 0 if the cartpole is balanced and satisfies $|x| < 0.05$, and $-1$ otherwise.

**(iii) Mountain Car:** The mountain car testbed [25] models a simulated car, which starts in a valley and must climb the hill to the right as quickly as possible. The task involves two states $[x, \dot{x}]$ and three policy parameters $[K_1, K_2, \sigma]$. Our control inputs for this problem are restricted to $\{-1, 1\}$. Our parameterized policy is given by $\pi(u_t = 1|x_t, \dot{x}_t) = P(K_1 sign(\dot{x}_t)\dot{x}_t^2 + K_2 + \epsilon_t < x_t)$, where $\epsilon_t \sim N(0, \sigma)$. Our initial acceleration is $f_0 = +1$; $f_{t+1} = u_t f_t$. The dynamics are given by $\dot{x}_{t+1} = \dot{x}_t + 0.001 f_t - 0.0025 \cos(3(x_t - 0.5))$, and $x_{t+1} = x_t + \dot{x}_t$

We run for 200 time steps, though the episode terminates once the mountaincar reaches its target at $x = 1.0$. The reward function is 0 if the car is at its target and $-1$ otherwise.

**(iv) Acrobot:** The acrobot [25] is a robot with 2 rotational links connected by an actuated motor. It has four states $[\theta_1, \dot{\theta}_1, \theta_2, \dot{\theta}_2]$ and parameters $K = [K_1, \ldots, K_8, \sigma]$. The acrobot is initialized to be close to $[\pi, 0, 0, 0]$ (pointing straight up), and the goal is to keep the acrobot balanced upright for as long as possible. Our control input is drawn from the policy $u \sim N(Lx + K^\top \phi(x), \sigma)$. Here $L$ is the optimal LQR controller for acrobot linearized around the stationary point, and $\phi(x) = [(\pi - \theta_1)\theta_2, \dot{\theta}_1\dot{\theta}_2, (\pi - \theta_1)\dot{\theta}_1, \theta_2\dot{\theta}_2, (\pi - \theta_1)|\pi - \theta_1|, \dot{\theta}_1|\dot{\theta}_1|, \theta_2|\theta_2|, \dot{\theta}_2|\dot{\theta}_2|]$. The dynamics are given by $\ddot{\theta}_1 = -\frac{d_2\ddot{\theta}_2 + \phi_1}{d_1}$, $\ddot{\theta}_2 = \frac{u + d_2/d_1\phi_1 - \phi_2}{m_2 l_{c2}^2 + I_2 - d_2^2/d_1}$, $d_1 = m_1 l_{c1}^2 + m_2(l_1^2 + l_{c2}^2 + 2l_1 l_{c2}\cos\theta_2) + I_1 + I_2$, $d_2 = m_2 * (l_{c2}^2 + l_1 * l_{c2}\cos\theta_2) + I_2$, $\phi_1 = -m_2 l_1 l_{c2}\dot{\theta}_2^2 - \sin(\theta_2) - 2m_2 l_1 l_{c2}\dot{\theta}_2\dot{\theta}_1\sin(\theta_2) + (m_1 l_{c1} + m_2 l_1)g\cos(\theta_1 - \pi/2) + \phi_2$, and $\phi_2 = m_2 + l_{c2}g\cos(\theta_1 + \theta_2 - \pi/2)$. Here $m_1 = 1, m_2 = 1, l_1 = 1, l_2 = 2, l_{c1} = 0.5, l_{c2} = 1, I_1 = 0.0833, I_2 = 0.33, g = 9.81$. The control interval was $0.02s$. We solve the dynamics using a fourth order Runge-Kutta method. Each episode is run for 400 time steps, though the episode terminates once the acrobot has failed (defined as whenever the height of the second link $t = -\cos(\theta_1) - \cos(\theta_1 + \theta_3) < 0.5$). The reward function is $-2$ for every time step after the failure occurs, and $-(1 - (-cos(\theta_1) - cos(\theta_1 + \theta_2))/2)^2$ otherwise.

## Footnotes

[1]Any infinite horizon MDP with discounted rewards can be $\epsilon$-approximated by a finite horizon MDP, using a horizon $H_\epsilon = \lceil \log_\gamma(\epsilon(1-\gamma)/R_{\max})\rceil$, where $R_{\max} = \max_s |R(s)|$.

[2]Estimating the baseline from the same data as the other terms in Equation (5.1) results in a biased estimator. This is often done in policy gradient methods and we do so in our experiments. It is however possible to retain an unbiased estimate by data splitting, which could include averaging over resamplings.

[3]In practice, since we cannot always find the true optimum of $\widehat{U}$ within the ESS region, differences in step direction do affect policies that are sampled. Other step directions or policy improvement rules may be substituted for the finite difference gradient step. For example, we could follow the natural gradient direction, or use an EM-based policy update [14].

[4]Though the Armijo rule has its own free parameters to choose, performance is much less sensitive to these hyper-parameters. We use the same Armijo rule parameters for all of our experiments.

[5]We can extend standard likelihood ratio policy gradient methods to use the importance sampled expected return estimate. In our experience this approach yields results comparable to the best fixed hand-tuned step size for each problem—hence alleviating the need of these methods for tuning the step size.

[6]The difference in performance due to different estimation procedure for the scalar baseline has been observed to be so small that only one plot is shown rather than both in [1].

[7]We do not plot REINFORCE as our experiments indicate that GPOMDP outperforms REINFORCE on these testbeds, a fact consistent with existing literature [1].

[8]We followed standard formulations of the control policy for LQR and cartpole. All policies are designed as functions of a linear combination of the policy parameters and hand-selected features.

# References

[1] J. Peters, S. Vijayakumar, and S. Schaal. Natural actor-critic. In *Proceedings of the European Machine Learning Conference (ECML)*, 2005.

[2] T. Mori, Y. Nakamura, M. Sato, and S. Ishii. Reinforcement learning for cpg-driven biped robot. In *AAAI*, 2004.

[3] R. Tedrake, T. W. Zhang, and H.S. Seung. Learning to walk in 20 minutes. In *Proceedings of the Fourteenth Yale Workshop on Adaptive and Learning Systems*, 2005.

[4] N. Kohl and P. Stone. Policy gradient reinforcement learning for fast quadrupedal locomotion, 2004.

[5] J. Zico Kolter and Andrew Y. Ng. Learning omnidirectional path following using dimensionality reduction. *RSS*, 2007.

[6] P. Glynn. Likelihood Ratio Gradient Estimation: An Overview". In *Proceedings of the 1987 Winter Simulation Conference, Atlanta, GA*, 1987.

[7] R. J. Williams. Simple statistical gradient-following algorithms for connectionist reinforcement learning. *Machine Learning*, 8:23, 1992.

[8] J. Baxter and P. Bartlett. Direct gradient-based reinforcement learning. *Journal of Artificial Intelligence Research*, 1999.

[9] E. Greensmith, P. Bartlett, and J. Baxter. Variance reduction techniques for gradient estimates in reinforcement learning. *Journal of Machine Learning Research*, 2004.

[10] Leonid Peshkin and Christian R. Shelton. Learning from scarce experience. In *Proceedings of the Nineteenth International Conference on Machine Learning*, 2002.

[11] R. Sutton, D. McAllester, S. Singh, and Y. Mansour. Policy gradient methods for reinforcement learning. In *NIPS 13*, 2000.

[12] J. Peters and S. Schaal. Policy gradient methods for robotics. In *Proceedings of the IEEE International Conference on Intelligent Robotics Systems*, 2006.

[13] A. Kong, J. S. Liu, and W. H. Wong. Sequential imputations and Bayesian missing data problems. *Journal of American Statistics Association*, 89:278–288, 1994.

[14] Jens Kober and Jan Peters. Policy search for motor primitives in robotics. *NIPS*, 2008.

[15] Dimitri P. Bertsekas. *Nonlinear Programming*. Athena Scientific, 2004.

[16] Richard S. Sutton. Dyna, an integrated architecture for learning, planning, and reacting, 1991.

[17] Xi-Ren Cao. A basic formula for on-line policy-gradient algorithms. *IEEE Transactions on Automatic Control*, 50:696–699, 2005.

[18] Jan Peters and Stefan Schaal. Reinforcement learning by reward-weighted regression for operational space control. In *In: Proceedings of the International Conference on Machine Learning (ICML*, pages 745–750, 2007.

[19] Andrew Ng and Michael Jordan. Pegasus: A policy search method for large mdps and pomdps. In *In Proceedings of the Sixteenth Conference on Uncertainty in Artificial Intelligence*, pages 406–415, 2000.

[20] S. Amari. Natural gradient works efficiently in learning. *Neural Computation*, 10, 1998.

[21] S. Kakade. A natural policy gradient. In *Advances in Neural Information Processing Systems*, volume 14 of *26*, 2001.

[22] Nicolas Le Roux, Pierre-Antoine Manzagol, and Yoshua Bengio. Topmoumoute online natural gradient algorithm. *NIPS*, 2007.

[23] Jan Peters. *Machine Learning of Motor Skills for Robotics*. PhD thesis, University of Southern California, 2007.

[24] M. Riedmiller, J. Peters, and S. Schaal. Evaluation of policy gradient methods and variants on the cart-pole benchmark. In *IEEE International Symposium on Approximate Dynamic Programming and Reinforcement Learning*, 2007.

[25] R. S. Sutton and A. G. Barto. *Reinforcement Learning: An Introduction*. MIT Press, 1998.

